# PROBABILISTIC CHARACTERIZATION OF

# NEURAL MODEL COMPUTATIONS

Richard M. Golden [†]
University of Pittsburgh, Pittsburgh, Pa. 15260

## ABSTRACT

Information retrieval in a neural network is viewed as a procedure in which the network computes a "most probable" or MAP estimate of the unknown information. This viewpoint allows the class of probability distributions, P, the neural network can acquire to be explicitly specified. Learning algorithms for the neural network which search for the "most probable" member of P can then be designed. Statistical tests which decide if the "true" or environmental probability distribution is in P can also be developed. Example applications of the theory to the highly nonlinear back-propagation learning algorithm, and the networks of Hopfield and Anderson are discussed.

## INTRODUCTION

A connectionist system is a network of simple neuron-like computing elements which can store and retrieve information, and most importantly make generalizations. Using terminology suggested by Rumelhart & McClelland [1], the computing elements of a connectionist system are called *units,* and each unit is associated with a real number indicating its *activity level.* The activity level of a given unit in the system can also influence the activity level of another unit. The degree of influence between two such units is often characterized by a parameter of the system known as a *connection strength.* During the *information retrieval process* some subset of the units in the system are activated, and these units in turn activate neighboring units via the inter-unit connection strengths. The activation levels of the neighboring units are then interpreted as

---

[†] Correspondence should be addressed to the author at the Department of Psychology, Stanford University, Stanford, California, 94305, USA.

the retrieved information. During the *learning process*, the values of the inter-unit connection strengths in the system are slightly modified each time the units in the system become activated by incoming information.

## DERIVATION OF THE SUBJECTIVE PF

Smolensky [2] demonstrated how the class of possible probability distributions that could be represented by a Harmony theory neural network model can be derived from basic principles. Using a simple variation of the arguments made by Smolensky, a procedure for deriving the class of probability distributions associated with *any* connectionist system whose information retrieval dynamics can be summarized by an additive energy function is briefly sketched. A rigorous presentation of this proof may be found in Golden [3].

Let a sample space, $S_p$, be a subset of the activation pattern state space, $S_d$, for a particular neural network model. For notational convenience, define the term *probability function* (pf) to indicate a function that assigns numbers between zero and one to the elements of $S_p$. For discrete random variables, the pf is a probability mass function. For continuous random variables, the pf is a probability density function. Let a particular stationary stochastic environment be represented by the scalar-valued pf, $p_e(X)$, where $X$ is a particular activation pattern. The pf, $p_e(X)$, indicates the *relative frequency* of occurrence of activation pattern $X$ in the network model's environment. A second pf defined with respect to sample space $S_p$ also must be introduced. This probability function, $p_s(X)$, is called the network's subjective pf. The pf $p_s(X)$ is interpreted as the *network's belief* that $X$ will occur in the network's environment.

The subjective pf may be derived by making the assumption that the information retrieval dynamical system, $D_s$, is optimal. That is, it is assumed that $D_s$ is an algorithm designed to transform a less probable state $X$ into a more probable state $X^*$ where the probability of a state is defined by the subjective pf $p_s(X;A)$, and where the elements of $A$ are the connection strengths among the units. Or in traditional engineering terminology, it is assumed that $D_s$ is a MAP (maximum a posteriori) estimation algorithm. The second assumption is that an *energy* function, $V(X)$, that is minimized by the system during the information retrieval process can be found with an *additivity* property. The additivity property says that if the neural network were partitioned into two physically unconnected subnetworks, then V(X) can be rewritten as $V_1(X_1) + V_2(X_2)$ where $V_1$ is the energy function minimized by the first subnetwork and $V_2$ is the energy function minimized by the second subnetwork. The third assumption is that V(X) provides a sufficient amount of information to specify the probability of activation pattern X. That is, $p_s(X) = G(V(X))$ where G is some continuous function. And the final assumption (following Smolensky[2]) is that statistical and physical independence are equivalent.

To derive $p_s(X)$, it is necessary to characterize G more specifically. Note that if probabilities are assigned to activation patterns such that physically independent substates of the system are also statistically independent, then the additivity property of V(X) forces G to be an exponential function since the only continuous function that maps addition into multiplication is the exponential[4]. After normalization and the assignment of unity to an irrelevant free parameter[2], the unique subjective pf for a network model that minimizes V(X) during the information retrieval process is:

$$p_s(X;A) = Z^{-1} exp[-V(X;A)] \tag{1}$$

$$Z = \int exp[-V(X;A)]dX \tag{2}$$

provided that $Z < C < \infty$. Note that the integral in (2) is taken over $S_p$. Also note that the pf, $p_s$, and sample space, $S_p$, specify a Markov Random Field since (1) is a Gibbs distribution[5].

*Example 1: Subjective pfs for associative back-propagation networks*

The information retrieval equation for an associative back-propagation[6] network can be written in the form $O=\Phi[I;A]$ where the elements of the vector O are the activity levels for the output units and the elements of the vector I are the activity levels for the input units. The parameter vector A specifies the values

of the "connection strengths" among the units in the system. The function $\Phi$ specifies the *architecture* of the network.

A natural additive energy function for the information retrieval dynamics of the least squares associative back-propagation algorithm is:

$$V(O) = |O - \Phi(I;A)|^2. \tag{3}$$

If $S_p$ is defined to be a real vector space such that $O \in S_p$, then direct substitution of V(O) for $V_d(X;A)$ into (1) and (2) yields a multivariate Gaussian density function with mean $\Phi(I;A)$ and covariance matrix equal to the identity matrix multiplied by 1/2. This multivariate Gaussian density function is $p_s(O|I;A)$. That is, with respect to $p_s(O|I;A)$, information retrieval in an associative back-propagation network involves retrieving the "most probable" output vector, O, for a given input vector, I.

*Example 2: Subjective pfs for Hopfield and BSB networks.*

The Hopfield [7] and BSB model [8,9] neural network models minimize the following energy function during information retrieval:

$$V(X) = -X^T M X \tag{4}$$

where the elements of X are the activation levels of the units in the system, and the elements of M are the connection strengths among the units. Thus, the subjective pf for these networks is:

$$p_S(\mathbf{X}) = Z^{-1} \, exp\,[\mathbf{X}^T \mathbf{M}\, \mathbf{X}] \quad where \quad Z = \sum exp\,[\mathbf{X}^T \mathbf{M}\, \mathbf{X}] \tag{5}$$

where the summation is taken over $S_p$.

## APPLICATIONS OF THE THEORY

If the subjective pf for a given connectionist system is known, then traditional analyses from the theory of statistical inference are immediately applicable. In this section some examples of how these analyses can aid in the design and analysis of neural networks are provided.

### Evaluating Learning Algorithms

Learning in a neural network model involves searching for a set of connection strengths or parameters that obtain a global minimum of a *learning* energy function. The theory proposed here explicitly shows how an optimal learning energy function can be constructed using the model's subjective pf and the environmental pf. In particular, optimal learning is defined as searching for the *most probable* connection strengths, given some set of observations (samples) drawn from the environmental pf. Given some mild restrictions upon the form of the a priori pf associated with the connection strengths, and for a sufficiently large set of observations, estimating the most probable connection strengths (MAP estimation) is equivalent to maximum likelihood estimation [10]

A well-known result [11] is that if the parameters of the subjective pf are represented by the parameter vector **A**, then the maximum likelihood estimate of **A** is obtained by finding the **A**\* that minimizes the function:

$$E(\mathbf{A}) = - <LOG \, [p_s(\mathbf{X};\mathbf{A})]> \tag{6}$$

where $< \, >$ is the expectation operator taken with respect to the environmental pf. Also note that (6) is the Kullback-Leibler [12] distance measure plus an irrelevant constant. Asymptotically, $E(\mathbf{A})$ is the logarithm of the probability of $\mathbf{A}$ given some set of observations drawn from the environmental pf.

Equation (6) is an important equation since it can aid in the evaluation and design of optimal learning algorithms. Substitution of the multivariate Gaussian associated with (3) into (6) shows that the back-propagation algorithm is doing gradient descent upon the function in (6). On the other hand, substitution of (5) into (6) shows that the Hebbian and Widrow-Hoff learning rules proposed for the Hopfield and BSB model networks are not doing gradient descent upon (6).

*Evaluating Network Architectures*

The global minimum of (6) occurs if and only if the subjective and environmental pfs are equivalent [12]. Thus, one crucial issue is whether *any* set of connection strengths exists such that the neural network's subjective pf can be made equivalent to a given environmental pf. If no such set of connection strengths exists, the subjective pf, $p_s$, is defined to be *misspecified*. White [11] and Lancaster [13] have introduced a statistical test designed to reject the null hypothesis that the subjective pf, $p_s$, is not misspecified. Golden [3] suggests a version of this test that is suitable for subjective pfs with many parameters.

## REFERENCES

1. D. E. Rumelhart, J. L. McClelland, and the PDP Research Group, Parallel distributed processing: Explorations in the microstructure of cognition, *1*, (MIT Press, Cambridge, 1986).
2. P. Smolensky, In D. E. Rumelhart, J. L. McClelland and the PDP Research Group (Eds.), Parallel distributed processing: Explorations in the microstructure of cognition, *1*, (MIT Press, Cambridge, 1986), pp. 194-281.

3. R. M. Golden, A unified framework for connectionist systems. Unpublished manuscript.

4. C. Goffman, Introduction to real analysis. (Harper and Row, N. Y., 1966), p. 65.

5. J. L. Marroquin, Probabilistic solution of inverse problems. A.I. Memo 860, MIT Press (1985).

6. D. E. Rumelhart, G. E. Hinton, & R. J. Williams, In D. E. Rumelhart, J. L. McClelland, and the PDP Research Group (Eds.), Parallel distributed processing: Explorations in the microstructure of cognition, *1*, (MIT Press, Cambridge, 1986), pp. 318-362.

7. J. J. Hopfield, Proceedings of the National Academy of Sciences, USA, *79*, 2554-2558 (1982).

8. J. A. Anderson, R. M. Golden, & G. L. Murphy, In H. Szu (Ed.), Optical and Hybrid Computing, SPIE, *634*, 260-276 (1986).

9. R. M. Golden, Journal of Mathematical Psychology, *30*, 73-80 (1986).

10. H. L. Van Trees, Detection, estimation, and modulation theory. (Wiley, N. Y., 1968).

11. H. White, Econometrica, *50*, 1-25 (1982).

12. S. Kullback & R. A. Leibler, Annals of Mathematical Statistics, *22*, 79-86 (1951).

13. T. Lancaster, Econometrica, *52*, 1051-1053 (1984).

## ACKNOWLEDGEMENTS

This research was supported in part by the Mellon foundation while the author was an Andrew Mellon Fellow in the Psychology Department at the University of Pittsburgh, and partly by the Office of Naval Research under Contract No. N-0014-86-K-0107 to Walter Schneider. This manuscript was revised while the author was an NIH postdoctoral scholar at Stanford University. This research was also supported in part by grants from the Office of Naval Research (Contract No. N00014-87-K-0671), and the System Development Foundation to David Rumelhart. I am very grateful to Dean C. Mumme for comments, criticisms, and helpful discussions concerning an earlier version of this manuscript. I would also like to thank David B. Cooper of Brown University for his suggestion that many neural network models might be viewed within a unified statistical framework.
